# Nonlinear Markov Networks for Continuous Variables

**Reimar Hofmann and Volker Tresp***
Siemens AG, Corporate Technology
Information and Communications
81730 München, Germany

## Abstract

We address the problem of learning structure in nonlinear Markov networks with continuous variables. This can be viewed as non-Gaussian multidimensional density estimation exploiting certain conditional independencies in the variables. Markov networks are a graphical way of describing conditional independencies well suited to model relationships which do not exhibit a natural causal ordering. We use neural network structures to model the quantitative relationships between variables. The main focus in this paper will be on learning the structure for the purpose of gaining insight into the underlying process. Using two data sets we show that interesting structures can be found using our approach. Inference will be briefly addressed.

## 1 Introduction

Knowledge about independence or conditional independence between variables is most helpful in "understanding" a domain. An intuitive representation of independencies is achieved by graphical models in which independency statements can be extracted from the structure of the graph. The two most popular types of graphical stochastical models are Bayesian networks which use a directed graph, and Markov networks which use an undirected graph. Whereas Bayesian networks are well suited to represent causal relationships, Markov networks are mostly used in cases where the user wants to express statistical correlation between variables. This is the case in image processing where the variables typically represent the grey levels of pixels and the graph encourages smoothness in the values of neighboring pixels (Markov random fields, Geman and Geman, 1984). We believe that Markov networks might be a useful representation in many domains where the concept of cause and effect is somewhat artificial. The learned structure of a Markov network also seems to be more easily communicated to non-experts; in a Bayesian network not all arc directions can be uniquely identified based on training data alone which makes a meaningful interpretation for the non-expert rather difficult.

As in Bayesian networks, direct dependencies between variables in Markov networks are represented by an arc between those variables and missing edges represent independencies (in Section 2 we will be more precise about the independencies represented in Markov networks). Whereas the graphical structure in Markov networks might be known a priori in some cases,

---
Reimar.Hofmann@mchp.siemens.de Volker.Tresp@mchp.siemens.de

the focus of this work is the case that structure is unknown and must be inferred from data. For both discrete variables and linear relationships between continuous variables algorithms for structure learning exist (Whittaker, 1990). Here we address the problem of learning structure for Markov networks of continuous variables where the relationships between variables are nonlinear. In particular we use neural networks for approximating the dependency between a variable and its Markov boundary. We demonstrate that structural learning can be achieved without a direct reference to a likelihood function and show how inference in such networks can be performed using Gibbs sampling. From a technical point of view, these *Markov boundary networks* perform multi-dimensional density estimation for a very general class of non-Gaussian densities.

In the next section we give a mathematical description of Markov networks and a formulation of the joint probability density as a product of compatibility functions. In Section 3.1 we discuss strucural learning in Markov networks based on a maximum likelihood approach and show that this approach is in general unfeasible. We then introduce our approach which is based on learning the Markov boundary of each variable. We also show how belief update can be performed using Gibbs sampling. In Section 4 we demonstrate that useful structures can be extraced from two data sets (Boston housing data, financial market) using our approach.

## 2   Markov Networks

The following brief introduction to Markov networks is adapted from Pearl (1988). Consider a strictly positive[1] joint probability density $p(x)$ over a set of variables $\mathcal{X} := \{x_1, \ldots, x_N\}$. For each variable $x_i$, let the *Markov boundary* of $x_i$, $\mathcal{B}_i \subseteq \mathcal{X} - \{x_i\}$, be the smallest set of variables that renders $x_i$ and $\mathcal{X} - (\{x_i\} \cup \mathcal{B}_i)$ independent under $p(x)$ (the Markov boundary is unique for strictly positive distributions). Let the *Markov network* $\mathcal{G}$ be the undirected graph with nodes $x_1, \ldots, x_N$ and edges between $x_i$ and $x_j$ if and only if $x_i \in \mathcal{B}_j$ (which also implies $x_j \in \mathcal{B}_i$). In other words, a Markov network is generated by connecting each node to the nodes in its Markov boundary. Then for any set $Z \subseteq (\mathcal{X} - \{x_i, x_j\})$, $x_i$ is independent of $x_j$ given $Z$ if and only if every path from $x_i$ to $x_j$ goes through at least one node in $Z$. In other words, two variables are independent if any path between those variables is "blocked" by a known variable. In particular a variable is independent of the remaining variables if the variables in its Markov boundary are known.

A *clique* in $G$ is a maximal fully connected subgraph. Given a Markov Network $G$ for $p(x)$ it can be shown that $p$ can be factorized as a product of *positive* functions on the cliques of $G$, i.e.

$$p(x) = \frac{1}{K} \prod_i g_i(x_{clique_i}) \tag{1}$$

where the product is over all cliques in the graph. $x_{clique_i}$ is the projection of $x$ to the variables of the $i$-th clique and the $g_i$ are the *compatibility functions* w.r.t. $clique_i$. $K = \int \prod_i g_i(x_{clique_i}) dx$ is the normalization constant. Note, that a state whose clique functions have large values has high probability. The theorem of Hammersley and Clifford states that the normalized product in equation 1 embodies all the conditional independencies portrayed by the graph (Pearl, 1988)[2] for any choice of the $g_i$.

If the graph is sparse, i.e. if many conditional independencies exist then the cliques might

be small and the product will be over low dimensional functions. Similar to Bayesian networks where the complexity of describing a joint probability density is greatly reduced by decomposing the joint density in a product of ideally low-dimensional conditional densities, equation 1 describes the decomposition of a joint probability density function into a product of ideally low-dimensional compatibility functions. It should be noted that Bayesian networks and Markov networks differ in which specific independencies they can represent (Pearl, 1988).

## 3 Learning the Markov Network

### 3.1 Likelihood Function Based Learning

Learning graphical stochastical models is usually decomposed into the problems of learning structure (that is the edges in the graph) and of learning the parameters of the joint density function under the constraint that it obeys the independence statements made by the graph. The idea is to generate candidate structures according to some search strategy, learn the parameters for this structure and then judge the structure on the basis of the (penalized) likelihood of the model or, in a fully Bayesian approach, using a Bayesian scoring metric.

Assume that the compatibility functions in equation 1 are approximated using a function approximator such as a neural network $g_i() \approx g_i^w(x)$. Let $\{x^p\}_{p=1}^N$ be a training set. With likelihood $L = \prod_{p=1}^N p^M(x^p)$ (where the $M$ in $p^M$ indicates a probability density *model* in contrast to the true distribution), the gradient of the log-likelihood with respect to weight $w_i$ in $g_i(.)$ becomes

$$\frac{\partial}{\partial w_i} \sum_{p=1}^N \log p^M(x^p) = \sum_{p=1}^N \frac{\partial}{\partial w_i} \log g_i^w(x_{clique_i}) - N \frac{\int (\frac{\partial}{\partial w_i} \log g_i^w(x_{clique_i})) \prod_j g_j^w(x_{clique_j}) dx}{\int \prod_j g_j^w(x_{clique_j}) dx}$$

(2)

where the sums are over $N$ training patterns. The gradient decomposes into two terms. Note, that only in the first term the training patterns appear explicitly and that, conveniently, the first term is only dependent on the clique $i$ which contains parameter $w_i$. The second term emerges from the normalization constant $K$ in equation 1. The difficulty is that the integrals in the second term can not be solved in closed form for universal types of compatibility functions $g_i$ and have to be approximated numerically, typically using a form of Monte Carlo integration. This is exactly what is done in the Boltzmann machine, which is a special case of a Markov network with discrete variables.[3]

Currently, we consider maximum likelihood learning based on the compatibility functions unsuitable, considering the complexity and slowness of Monte Carlo integration (i.e. stochastic sampling). Note, that for structural learning the maximum likelihood learning is in the inner loop and would have to be executed repeatedly for a large number of structures.

### 3.2 Markov Boundary Learning

The difficulties in using maximum likelihood learning for finding optimal structures motivated the approach pursued in this paper. If the underlying true probability density is known the structure in a Markov network can be found using either the *edge deletion method* or the

*Markov boundary method* (Pearl, 1988). The edge deletion method uses the fact that variables $a$ and $b$ are *not* connected by an edge if and only if $a$ and $b$ are independent given all other variables. Evaluating this test for each pair of variables reveals the structure of the network. The Markov boundary method consists of determining - for each variable $a$ - its Markov boundary and connecting $a$ to each variable in its Markov boundary. Both approaches are simple if we have a reliable test for true conditional independence.

Both methods cannot be applied directly for learning structure from data since here tests for conditional independence cannot be based on the true underlying probability distribution (which is unknown) but has to be inferred from a finite data set. The hope is that dependencies which are strong enough to be supported by the data can still be reliably identified. It is, however not difficult to construct cases where simply using an (unreliable) statistical test for conditional independence with the edge deletion method does not work well.[4]

We now describe our approach, which is motivated by the Markov boundary method. First, we start with a fully connected graph. We train a model $p_i^M$ to approximate the conditional density of each variable $i$, given the current candidate variables for its Markov boundary $\mathcal{B}_i'$ which initially are all other variables. For this we can use a wide variety of neural networks. We use conditional Parzen windows

$$p_i^M(x_i|x_{\mathcal{B}_i'}) = \frac{\sum_{p=1}^N G(x_{\{i\}\cup\mathcal{B}_i'}; x_{\{i\}\cup\mathcal{B}_i'}^p, \Sigma_i)}{\sum_{p=1}^N G(x_{\mathcal{B}_i'}; x_{\mathcal{B}_i'}^p, \Sigma_{i,\mathcal{B}_i'})}, \tag{3}$$

where $\{x^p\}_{p=1}^N$ is the training set and $G(x; \mu, \Sigma)$ is our notation for a multidimensional Gaussian centered at $\mu$ with covariance matrix $\Sigma$ evaluated at $x$. The Gaussians in the nominator are centered at $x_{\{i\}\cup\mathcal{B}_i'}^p$ which is the location of the $p$-th sample in the joint input/output ($\{x_i\}\cup\mathcal{B}_i'$) space and the Gaussians in the denominator are centered at $x_{\mathcal{B}_i'}^p$ which is the location of the $p$-th sample in the input space ($\mathcal{B}_i'$). There is one covariance matrix $\Sigma_i$ for each conditional density model which is shared between all the Gaussians in that model. $\Sigma_i$ is restricted to a diagonal matrix where the diagonal elements in all dimensions except the output dimension $i$, are the same. So there are only two free parameters in the matrix: The variance in the output dimension and the variance in all input dimensions. $\Sigma_{i,\mathcal{B}_i'}$ is equal to $\Sigma_i$ except that the row and column corresponding to the output dimension have been deleted. For each conditional model $p_i^M$, $\Sigma_i$ was optimized on the basis of the leave-one-out cross validation log-likelihood.

Our approach is based on tentatively removing edges from the model. Removing an edge decreases the size of the Markov boundary candidates of both affected variables and thus decreases the number of inputs in the corresponding two conditional density models. With the inputs removed, we retrain the two models (in our case, we simply find the optimal $\Sigma_i$ for the two conditional Parzen windows). If the removal of the edge was correct, the leave-one-out cross validation log-likelihood (*model-score*) of the two models should improve since an unnecessary input is removed. (Removing an unnecessary input typically decreases model variance.) We therefore remove an edge if the model-scores of both models improve. Let's define as *edge-removal-score* the smaller of the two improvements in model-score.

Here is the algorithm in pseudo code:

- Start with a fully connected network

- Until no edge-removal-score is positive:
    - for all edges $edge_{ij}$ in the network
        * calculate the model-scores of the reduced models $p_i^M(x_i|\mathcal{B}_i' - \{j\})$ and $p_i^M(x_j|\mathcal{B}_j' - \{i\})$
        * compare with the model-scores of the current models $p_i^M(x_i|\mathcal{B}_i')$ and $p_i^M(x_j|\mathcal{B}_j')$
        * set the edge-removal-score to the smaller of both model-score improvements
    - remove the edge for which the edge-removal-score is in maximum.

- end

### 3.3 Inference

Note that we have learned the structure of the Markov network without an explicit representation of the probability density. Although the conditional densities $p(x_i|\mathcal{B}_i)$ provide sufficient information to calculate the joint probability density the latter can not be easily computed. More precisely, the conditional densities overdetermine the joint density which might lead to problems if the conditional densities are estimated from data. For inference, we are typically interested in the expected value of an unknown variable, given an arbitrary set of known variables, which can be calculated using Gibbs sampling. Note, that the conditional densities $p^M(x_i|\mathcal{B}_i)$ which are required for Gibbs sampling are explicitly modeled in our approach by the conditional Parzen windows. Also note, that sampling from the conditional Parzen model (as well as many other neural networks, such as mixture of experts models) is easy.[5] In Hofmann (1997) we show that Gibbs sampling from the conditional Parzen models gives significantly better results than running inference using either a kernel estimator or a Gaussian mixture model of the joint density.

## 4 Experiments

In our first experiment we used the Boston housing data set, which contains 506 samples. Each sample consists of the housing price and 13 other variables which supposedly influence the housing price in a Boston neighborhood. Maximizing the cross validation log-likelihood as score as described in the previous chapters results in a Markov network with 68 edges.

While cross validation gives an unbiased estimate of whether a direct dependency exists between two variables the estimate can have a large variance depending on the size of the given data set. If the goal of the experiment is to interpret the resulting structure one would prefer to see only those edges corresponding to direct dependencies which can be clearly identified from the given data set. In other words, if the relationship between two variables observed on the given data set is so weak that we can not be sure that it is not just an effect of the finite data set size, then we do not want to display the corresponding edge. This can be achieved by adding a penalty per edge to the score of the conditional density models. (figure 1).

Figure 2 shows the resulting Markov network for a penalty per edge of 0.2. The goal of the original experiment for which the Boston housing data were collected was to examine whether the air quality (5) has direct influence on the housing price (14). Our algorithm did not find such an influence - in accordance with the original study. It found that the percentage of low status population (13) and the average number of rooms (6) are in direct relationship with the housing price. The pairwise relationships between these three variables are displayed in figure 3.

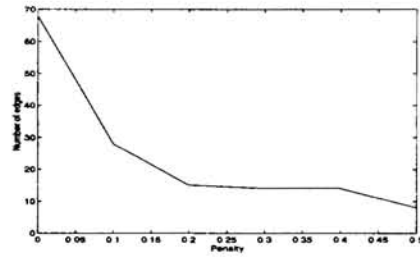

Figure 1: Number of edges in the Markov network for the Boston housing data as a function of the penalty per edge.

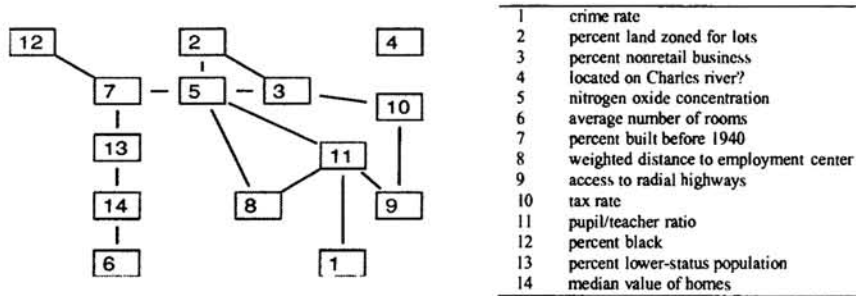

| | |
|---|---|
| 1 | crime rate |
| 2 | percent land zoned for lots |
| 3 | percent nonretail business |
| 4 | located on Charles river? |
| 5 | nitrogen oxide concentration |
| 6 | average number of rooms |
| 7 | percent built before 1940 |
| 8 | weighted distance to employment center |
| 9 | access to radial highways |
| 10 | tax rate |
| 11 | pupil/teacher ratio |
| 12 | percent black |
| 13 | percent lower-status population |
| 14 | median value of homes |

Figure 2: Final structure of a run on the full Boston housing data set (penalty = 0.2).

The scatter plots visualize the relationship between variables 13 and 14, 6 and 14 and between 6 and 13 (from left to right). The left and the middle correspond to edges in the Markov network whereas for the right diagram the corresponding edge (6-13) is missing even though both variables are clearly dependent. The reason is, that the dependency between 6 and 13 can be explained as indirect relationship via variable 14. The Markov network tells us that 13 and 6 are independent given 14, but dependent if 14 is unknown.

In a second experiment we used a financial dataset. Each pattern corresponds to one business day. The variables in our model are relative changes in certain economic variables from the last business day to the present day which were expected to possibly influence the development of the German stock index DAX and the composite DAX, which contains a larger selection of stocks than the DAX. We used 500 training patterns consisting of 12 variables (figure 4). In comparison to the Boston housing data set most relationships are very weak. Using a penalty per edge of 0.2 leads to a very sparse model with only three edges (2-12, 12-1,5-11) (not shown). A penalty of 0.025 results in the model shown in figure 4. Note, that the composite

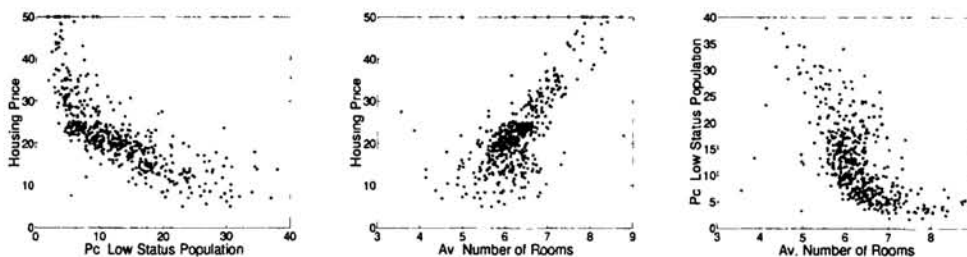

Figure 3: Pairwise relationship between the variables 6, 13 and 14. Displayed are all data points in the Boston housing data set.

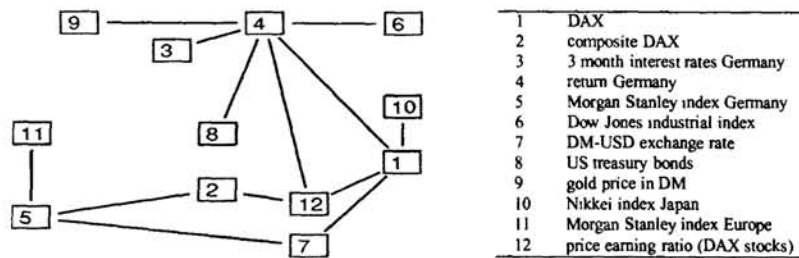

Figure 4: Final structure of a run on the financial data set with a penalty of 0.025. The small numbers next to the edges indicate the strength of the connection, i.e. the decrease in score (excluding the penalty) when the edge is removed. All variables are relative changes - not absolute values.

DAX is connected to the DAX mainly through the price earning ratio. While the DAX has direct connections to the Nikkei index and to the DM-USD exchange rate the composite DAX has a direct connection to the Morgan Stanley index for Germany. Recall, that composite DAX contains the stocks of many smaller companies in addition to the DAX stocks. The graph structure might be interpreted (with all caution) in the way that the composite DAX (including small companies) has a stronger dependency on national business whereas the DAX (only including the stock of major companies) reacts more to international indicators.

## 5 Conclusions

We have demonstrated, to our knowledge for the first time, how nonlinear Markov networks can be learned for continuous variables and we have shown that the resulting structures can give interesting insights into the underlying process. We used a representation based on models of the conditional probability density of each variable given its Markov boundary. These models can be trained locally. We showed how searching in the space of all possible structures can be done using this representation.

We suggest to use the conditional densities of each variable given its Markov boundary also for inference by Gibbs sampling. Since the required conditional densities are modeled explicitly by our approach and sampling from these is easy, Gibbs sampling is easier and faster to realize than with a direct representation of the joint density.

A topic of further research is the variance in resulting structures, i.e. the fact that different structures can lead to almost equally good models. It would for example be desirable to indicate to the user in a principled way the certainty of the existence or nonexistence of edges.

## Footnotes

[1] To simplify the discussion we will assume strict positivity for the rest of this paper. For some of the statements weaker conditions may also be sufficient. Note that strict positivity implies that functional constraints (for example, $a = b$) are excluded.

[2] In terms of graphical models: The graph G is an I-map of p.

[3] A fully connected Boltzmann machine does not display any independencies and we only have one clique consisting of all variables. The compatibility function is $g() = \exp(-\sum w_{ij} s_i s_j)$. The Boltzmann machine typically contains hidden variables, such that not only the second term (corresponding to the unclamped phase) in equation 2 has to be approximated using stochastic sampling but also the first term. (In this paper we only consider the case that data are complete).

[4]The problem is that in the edge deletion method the decision is made independently for each edge whether or not it should be present. There are however cases where it is obvious that at least one of two edges must be present although the edge deletion method which tests each edge individually removes both.

[5] Readers not familiar with Gibbs sampling, please consult Geman and Geman (1984).

## References

Geman, S., and Geman, D. (1984). Stochastic relaxations, Gibbs distributions and the Bayesian restoration of images. *IEEE Trans. on Pattern Analysis and Machine Intelligence* PAMI-6 (no. 6):721-42

Hofmann, R. (1997). *Inference in Markov Blanket Models*. Technical report, in preparation.

Monti, S., and Cooper, G. (1997). Learning Bayesian belief networks with neural network estimators. In *Neural Information Processing Systems 9.*, MIT Press.

Pearl, J. (1988). *Probabilistic reasoning in intelligent systems*. San Mateo: Morgan Kaufmann.

Whittaker, J. (1990). *Graphical models in applied multivariate statistics*. Chichester, UK: John Wiley and Sons.
